# MELONET I: Neural Nets for Inventing Baroque-Style Chorale Variations

**Dominik Hörnel**
dominik@ira.uka.de
Institut für Logik, Komplexität und Deduktionssysteme
Universität Fridericiana Karlsruhe (TH)
Am Fasanengarten 5
D–76128 Karlsruhe, Germany

## Abstract

MELONET I is a multi-scale neural network system producing baroque-style melodic variations. Given a melody, the system invents a four-part chorale harmonization and a variation of any chorale voice, after being trained on music pieces of composers like J. S. Bach and J. Pachelbel. Unlike earlier approaches to the learning of melodic structure, the system is able to learn and reproduce high-order structure like harmonic, motif and phrase structure in melodic sequences. This is achieved by using mutually interacting feedforward networks operating at different time scales, in combination with Kohonen networks to classify and recognize musical structure. The results are chorale partitas in the style of J. Pachelbel. Their quality has been judged by experts to be comparable to improvisations invented by an experienced human organist.

## 1 INTRODUCTION

The investigation of neural information structures in music is a rather new, exciting research area bringing together different disciplines such as computer science, mathematics, musicology and cognitive science. One of its aims is to find out what determines the personal style of a composer. It has been shown that neural network models – better than other AI approaches – are able to learn and reproduce style-dependent features from given examples, e.g., chorale harmonizations in the style of Johann Sebastian Bach (Hild et al., 1992). However when dealing with *melodic* sequences, e.g., folk-song style melodies, all of these models have considerable difficulties to learn even simple structures. The reason is that they are unable to capture high-order structure such as harmonies, motifs and phrases simultaneously occurring at multiple time scales. To overcome this problem, Mozer (Mozer, 1994)

proposes context units that learn reduced descriptions of a sequence of individual notes. A similar approach in MELONET (Feulner et Hörnel, 1994) uses *delayed update units* that do not fire each time their input changes but rather at discrete time intervals. Although these models perform well on artificial sequences, they produce melodies that suffer from a lack of global coherence.

The art of melodic variation has a long tradition in Western music. Almost every great composer has written music pieces inventing variations of a given melody, e.g., Mozart's famous variations KV 265 on the melody "Ah! Vous dirai-je, Maman", also known as "Twinkle twinkle little star". At the beginning of this tradition there is the baroque type of chorale variations. These are organ or harpsichord variations of a chorale melody composed for use in the Protestant church. A prominent representative of this kind of composition is J. Pachelbel (1653 - 1706) who wrote about 50 chorale variations or partitas on various chorale melodies.

## 2   TASK DESCRIPTION

Given a chorale melody, the learning task is achieved in two steps:

1. A chorale harmonization of the melody is invented.
2. One of the voices of the resulting chorale is chosen and provided with melodic variations.

Both subtasks are directly learned from music examples composed by J. Pachelbel and performed in an interactive composition process which results in a chorale variation of the given melody. The first task is performed by HARMONET, a neural network system which is able to harmonize melodies in the style of various composers like J. S. Bach. The second task is performed by the neural network system MELONET I, presented in the following. For simplicity we have considered melodic variations consisting of 4 sixteenth notes for each melody quarter note. This is the most common variation type used by baroque composers and presents a good starting point for even more complex variation types, since there are enough music examples for training and testing the networks, and because it allows the representation of higher-scale elements in a rather straightforward way.

HARMONET is a system producing four-part chorales in various harmonization styles, given a one-part melody. It solves a musical real-world problem on a performance level appropriate for musical practice. Its power is based on a coding scheme capturing musically relevant information, and on the integration of neural networks and symbolic algorithms in a hierarchical system, combining the advantages of both. The details are not discussed in this paper. See (Hild et al., 1992) or (Hörnel et Ragg, 1996a) for a detailed account.

## 3   A MULTI-SCALE NEURAL NETWORK MODEL

The learning goal is twofold. On the one hand, the results produced by the system should conform to musical rules. These are melodic and harmonic constraints such as the correct resolving of dissonances or the appropriate use of successive interval leaps. On the other hand, the system should be able to capture stilistic features from the learning examples, e.g., melodic shapes preferred by J. Pachelbel. The observation of musical rules and the aesthetic conformance to the learning set can be achieved by a multi-scale neural network model. The complexity of the learning task is reduced by decomposition in three subtasks (see Figure 1):

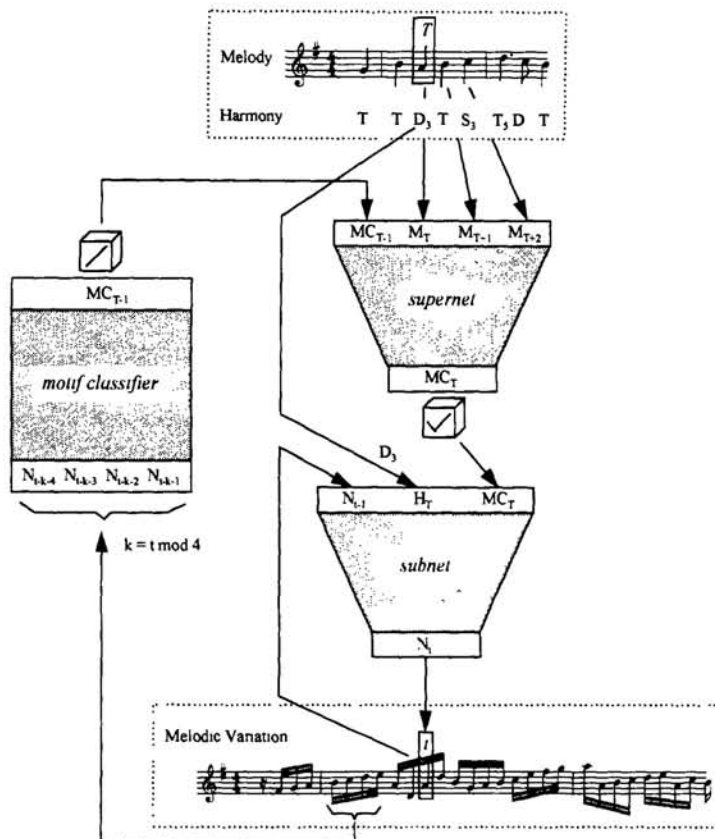

Figure 1: Structure of the system and process of composing a new melodic variation. A melody (previously harmonized by HARMONET) is passed to the supernet which predicts the current motif class $MC_T$ from a local window given by melody notes $M_T$ to $M_{T+2}$ and preceding motif class $MC_{T-1}$. A similar procedure is performed at a lower time scale by the subnet which predicts the next motif note $N_t$ based on $MC_T$, current harmony $H_T$ and preceding motif note $N_{t-1}$. The result is then returned to the supernet through the motif classifier to be considered when computing the next motif class $MC_{T+1}$.

1. A melody variation is considered at a higher time scale as a sequence of melodic groups, so-called *motifs*. Each quarter note of the given melody is varied by one motif. Before training the networks, motifs are classified according to their similarity.

2. One neural network is used to learn the abstract sequence of motif classes. Motif classes are represented in a 1-of-n coding form where n is a fixed number of classes. The question it solves is: What kind of motif 'fits' a melody note depending on melodic context and the motif that has occurred before? No concrete notes are fixed by this network. It works at a higher scale and will therefore be called *supernet* in the following.

3. Another neural network learns the implementation of abstract motif classes into concrete notes depending on a given harmonic context. It produces a sequence of sixteenth notes – four notes per motif – that result in a melodic variation of the given melody. Because it works one scale below the supernet, it is called *subnet*.

4. The subnet sometimes invents a sequence of notes that does not coincide

with the motif class determined by the supernet. This motif will be considered when computing the next motif class, however, and should therefore match the notes previously formed by the subnet. It is therefore reclassified by the *motif classifier* before the supernet determines the next motif class.

The motivation of this separation into supernet and subnet arised from the following consideration: Having a neural network that learns sequences of sixteenth notes, it would be easier for this network to predict notes given a *contour* of each motif, i.e. a sequence of interval directions to be produced for each quarter note. Consider a human organist who improvises a melodic variation of a given melody in real time. Because he has to take his decisions in a fraction of a second, he must at least have some rough idea in mind about what kind of melodic variation should be applied to the next melody note to obtain a meaningful continuation of the variation. Therefore, a neural network was introduced at a higher time scale, the training of which really improved the overall behavior of the system and not just shifted the learning problem to another time scale.

# 4   MOTIF CLASSIFICATION AND RECOGNITION

In order to realize learning at different time scales as described above, we need a recognition component to find a suitable classification of motifs. This can be achieved using unsupervised learning, e.g., *agglomerative hierarchical clustering* or Kohonen's *topological feature maps* (Kohonen, 1990). The former has the disadvantage however that an appropriate distance measure is needed which determines the similarity between small sequences of notes respectively intervals, whereas the latter allows to obtain appropriate motif classes through self-organization within a two-dimensional surface. Figure 2 displays the motif representation and distribution of motif contours over a 10x10 Kohonen feature map. In MELONET I, the Kohonen algorithm is applied to all motifs contained in the training set. Afterwards a corresponding motif classification tree is recursively built from the Kohonen map. While cutting this classification tree at lower levels we can get more and more classes. One important problem remains to find an appropriate number of classes for the given learning task. This will be discussed in section 6.

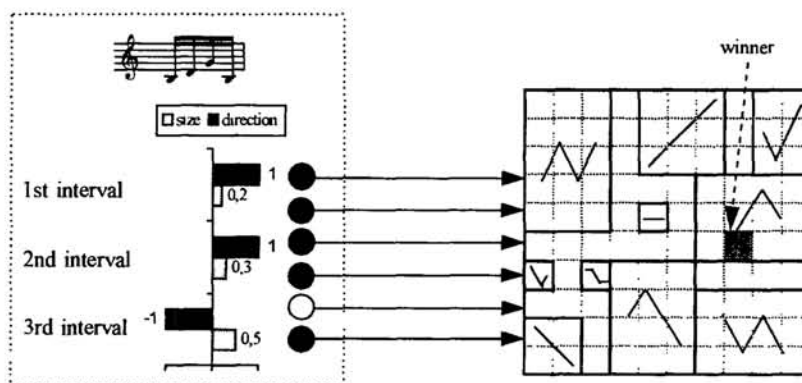

Figure 2: Motif representation example (left) and motif contour distribution (right) over a 10x10 Kohonen feature map developed from one Pachelbel chorale variation (initial update area 6x6, initial adaptation height 0.95, decrease factor 0.995). Each cell corresponds to one unit in the KFM. One can see the arrangement of regions responding to motifs having different motif contours.

# 5 REPRESENTATION

In general one can distinguish two groups of motifs: *Melodic motifs* prefer small intervals, mainly seconds, *harmonic motifs* prefer leaps and harmonizing notes (chord notes). Both motif groups heavily rely on harmonic information. In melodic motifs dissonances should be correctly resolved, in harmonic motifs notes must fit the given harmony. Small deviations may have a significant effect on the quality of musical results. Thus our idea was to integrate musical knowledge about interval and harmonic relationships into an appropriate *interval representation*. Each note is represented by its interval to the first motif note, the so-called *reference note*. This is an important element contributing to the success of MELONET I. A similar idea for Jazz improvisation was followed in (Baggi, 1992).

The interval coding shown in Table 1 considers several important relationships: *neighboring* intervals are realized by overlapping bits, *octave invariance* is represented using a special octave bit. The activation of the overlapping bit was reduced from 1 to 0.5 in order to allow a better distinction of the intervals. 3 bits are used to distinguish the direction of the interval, 7 bits represent interval size. Complementary intervals such as ascending thirds and descending sixths have similar representations because they lead to the same note and can therefore be regarded as *harmonically equivalent*. A simple rhythmic element was then added using a *tenuto* bit (not shown in Table 1) which is set when a note is tied to its predecessor. This final 3+1+7+1=12 bit coding gave the best results in our simulations.

Table 1: Complementary Interval Coding

| | direction | | | octave | interval size | | | | | | |
|---|---|---|---|---|---|---|---|---|---|---|---|
| ninth ↘ | 1 | 0 | 0 | 1 | 0 | 0 | 0 | 0 | 0 | 0.5 | 1 |
| octave ↘ | 1 | 0 | 0 | 1 | 1 | 0 | 0 | 0 | 0 | 0 | 0.5 |
| seventh ↘ | 1 | 0 | 0 | 0 | 0.5 | 1 | 0 | 0 | 0 | 0 | 0 |
| sixth ↘ | 1 | 0 | 0 | 0 | 0 | 0.5 | 1 | 0 | 0 | 0 | 0 |
| fifth ↘ | 1 | 0 | 0 | 0 | 0 | 0 | 0.5 | 1 | 0 | 0 | 0 |
| fourth ↘ | 1 | 0 | 0 | 0 | 0 | 0 | 0 | 0.5 | 1 | 0 | 0 |
| third ↘ | 1 | 0 | 0 | 0 | 0 | 0 | 0 | 0 | 0.5 | 1 | 0 |
| second ↘ | 1 | 0 | 0 | 0 | 0 | 0 | 0 | 0 | 0 | 0.5 | 1 |
| prime → | 0 | 1 | 0 | 0 | 1 | 0 | 0 | 0 | 0 | 0 | 0.5 |
| second ↗ | 0 | 0 | 1 | 0 | 0.5 | 1 | 0 | 0 | 0 | 0 | 0 |
| third ↗ | 0 | 0 | 1 | 0 | 0 | 0.5 | 1 | 0 | 0 | 0 | 0 |
| fourth ↗ | 0 | 0 | 1 | 0 | 0 | 0 | 0.5 | 1 | 0 | 0 | 0 |
| fifth ↗ | 0 | 0 | 1 | 0 | 0 | 0 | 0 | 0.5 | 1 | 0 | 0 |
| sixth ↗ | 0 | 0 | 1 | 0 | 0 | 0 | 0 | 0 | 0.5 | 1 | 0 |
| seventh ↗ | 0 | 0 | 1 | 0 | 0 | 0 | 0 | 0 | 0 | 0.5 | 1 |
| octave ↗ | 0 | 0 | 1 | 1 | 1 | 0 | 0 | 0 | 0 | 0 | 0.5 |
| ninth ↗ | 0 | 0 | 1 | 1 | 0.5 | 1 | 0 | 0 | 0 | 0 | 0 |

Now we still need a representation for harmony. It can be encoded as a *harmonic field* which is a vector of chord notes of the diatonic scale. The tonic T in C major for example contains 3 chord notes – C, E and G – which correspond to the first, third and fifth degree of the C major scale (1010100). This representation may be further improved. We have already mentioned that each note is represented by the interval to the *first* motif note (reference note). We can now encode the harmonic field starting with the first motif note instead of the first degree of the scale. This is equivalent to rotating the bits of the harmonic field vector. An example is displayed in Figure 3. The harmony of the motif is the dominant D, the first motif note is B which corresponds to the seventh degree of the C major scale. Therefore the

harmonic field for D (0100101) is rotated by one position to the right resulting in (1010010). Starting with the first note B, the harmonic field indicates the intervals that lead to harmonizing notes B, D and G. In the right part of Figure 3 one can see a correspondance between bits activated in the harmonic field and bits set to 1 in the three interval codings. This kind of representation helps the neural network to directly establish a relationship between intervals and given harmony.



| | third up | 0 | 0 | 1 | 0 | 0 | 0.5 | 1 | 0 | 0 | 0 | 0 |
| | sixth up | 0 | 0 | 1 | 0 | 0 | 0 | 0 | 0 | 0.5 | 1 | 0 |
| | prime | 0 | 1 | 0 | 0 | 1 | 0 | 0 | 0 | 0 | 0 | 0.5 |
| D | harmonic field | | | | | 1 | 0 | 1 | 0 | 0 | 1 | 0 |

Figure 3: Example illustrating the relationship between interval coding and rotated harmonic field. Each note is represented by its interval to the first note.

## 6  PERFORMANCE

We carried out several simulations to evaluate the performance of the system. Many improvements could be found however by just listening to the improvisations produced by the neural organist. One important problem was to find an appropriate number of classes for the given learning task. The following table lists the classification rate on the learning and validation set of the supernet and the subnet using 5, 12 and 20 motif classes. The learning set was automatically built from 12 Pachelbel chorale variations corresponding to 2220 patterns for the subnet and 555 for the supernet. The validation set includes 6 Pachelbel variations corresponding to 1396 patterns for the subnet and 349 for the supernet. Supernet and subnet were then trained independently with the RPROP learning algorithm.

| | supernet | | | subnet | | |
| | 5 classes | 12 classes | 20 classes | 5 classes | 12 classes | 20 classes |
|---|---|---|---|---|---|---|
| learning set | 91.17% | 86.85% | 87.57% | 86.31% | 93.92% | 95.68% |
| validation set | 49.85% | 40.69% | 37.54% | 79.15% | 83.38% | 86.96% |

The classification rate of both networks strongly depends on the number of classes, esp. on the validation set of the supernet. The smaller the number of classes, the better is the classification of the supernet because there are less alternatives to choose from. We can also notice an opposite development of the classification behavior for the subnet. The bigger the number of classes, the easier the subnet will be able to determine concrete motif notes for a given motif class. One can imagine that the optimal number of classes lies somewhere in the middle. Another idea is to form a committee of networks each of which is trained with different number of classes.

We have also tested MELONET I on melodies that do not belong to the baroque era. Figure 4 shows a harmonization and variation of the melody "Twinkle twinkle little star" used by Mozart in his famous piano variations. It was produced by a network committee formed by 3*2=6 networks trained with 5, 12 and 20 classes.

## 7  CONCLUSION

We have presented a neural network system inventing baroque-style variations on given melodies whose qualities are similar to those of an experienced human organ-

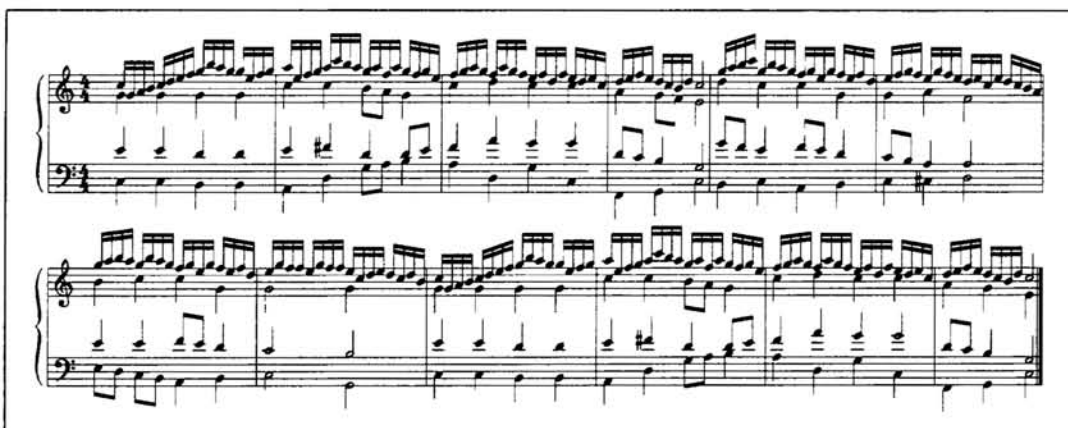

Figure 4: Melodic variation on "Twinkle twinkle little star"

ist. The complex musical task could be learned introducing a multi-scale network model with two neural networks cooperating at different time scales, together with an unsupervised learning mechanism able to classify and recognize relevant musical structure.

We are about to test this multi-scale approach on learning examples of other epochs, e.g., on compositions of classical composers like Haydn and Mozart or on Jazz improvisations. First results confirm that the system is able to reproduce style-specific elements of other kinds of melodic variation as well. Another interesting question is whether the global coherence of the musical results may be further improved adding another network working at a higher level of abstraction, e.g., at a phrase level. In summary, we believe that this approach presents an important step towards the learning of complete melodies.

# References

Denis L. Baggi. *NeurSwing: An Intelligent Workbench for the Investigation of Swing in Jazz.* In: Readings in Computer-Generated Music, IEEE Computer Society Press, pp. 79-94, 1992.

Johannes Feulner, Dominik Hörnel. *MELONET: Neural networks that learn harmony-based melodic variations.* In: Proceedings of the 1994 International Computer Music Conference. ICMA Arhus, pp. 121-124, 1994.

Hermann Hild, Johannes Feulner, Wolfram Menzel. *HARMONET: A Neural Net for Harmonizing Chorales in the Style of J. S. Bach.* In: Advances in Neural Information Processing 4 (NIPS 4), pp. 267-274. 1992.

Dominik Hörnel, Thomas Ragg. *Learning Musical Structure and Style by Recognition, Prediction and Evolution.* In: Proceedings of the 1996 International Computer Music Conference. ICMA Hong Kong, pp. 59-62, 1996.

Dominik Hörnel, Thomas Ragg. *A Connectionist Model for the Evolution of Styles of Harmonization.* In: Proceedings of the 1996 International Conference on Music Perception and Cognition. Montreal, 1996.

Teuvo Kohonen. *The Self-Organizing Map.* In: Proceedings of the IEEE, Vol. 78, no. 9, pp. 1464-1480, 1990.

Michael C. Mozer. *Neural Network music composition by prediction.* In: Connection Science 6(2,3), pp. 247-280, 1994.
